# Computing with Almost Optimal Size Neural Networks

**Kai-Yeung Siu**
Dept. of Electrical & Comp. Engineering
University of California, Irvine
Irvine, CA 92717

**Vwani Roychowdhury**
School of Electrical Engineering
Purdue University
West Lafayette, IN 47907

**Thomas Kailath**
Information Systems Laboratory
Stanford University
Stanford, CA 94305

## Abstract

Artificial neural networks are comprised of an interconnected collection of certain nonlinear devices; examples of commonly used devices include linear threshold elements, sigmoidal elements and radial-basis elements. We employ results from harmonic analysis and the theory of rational approximation to obtain almost tight lower bounds on the size (i.e. number of elements) of neural networks. The class of neural networks to which our techniques can be applied is quite general; it includes any feedforward network in which each element can be piecewise approximated by a low degree rational function. For example, we prove that any depth-$(d + 1)$ network of sigmoidal units or linear threshold elements computing the parity function of $n$ variables must have $\Omega(dn^{1/d-\epsilon})$ size, for any fixed $\epsilon > 0$. In addition, we prove that this lower bound is almost tight by showing that the parity function can be computed with $O(dn^{1/d})$ sigmoidal units or linear threshold elements in a depth-$(d + 1)$ network. These almost tight bounds are the first known complexity results on the size of neural networks with depth more than two. Our lower bound techniques yield a unified approach to the complexity analysis of various models of neural networks with feedforward structures. Moreover, our results indicate that in the context of computing highly oscillating symmetric Boolean func-

tions, networks of continuous-output units such as sigmoidal elements do not offer significant reduction in size compared with networks of linear threshold elements of binary outputs.

# 1    Introduction

Recently, artificial neural networks have found wide applications in many areas that require solutions to nonlinear problems. One reason for such success is the existence of good "learning" or "training" algorithms such as Backpropagation [13] that provide solutions to many problems for which traditional attacks have failed. At a more fundamental level, the computational power of neural networks comes from the fact that each basic processing element computes a nonlinear function of its inputs. Networks of these nonlinear elements can yield solutions to highly complex and nonlinear problems. On the other hand, because of the nonlinear features, it is very difficult to study the fundamental limitations and capabilities of neural networks. Undoubtedly, any significant progress in the applications of neural networks must require a deeper understanding of their computational properties.

We employ classical tools such as harmonic analysis and rational approximation to derive new results on the computational complexity of neural networks. The class of neural networks to which our techniques can be applied is quite large; it includes feedforward networks of sigmoidal elements, linear threshold elements, and more generally, elements that can be piecewise approximated by low degree rational functions.

## 1.1    Background, Related Work and Definitions

A widely accepted model of neural networks is the feedforward multilayer network in which the basic processing element is a *sigmoidal element*. A sigmoidal element computes a function $f(X)$ of its input variables $X = (x_1, \ldots, x_n)$ such that

$$f(X) = \sigma(F(X)) = \frac{2}{1 + e^{-F(X)}} - 1 = \frac{1 - e^{-F(X)}}{1 + e^{-F(X)}}$$

where

$$F(X) = \sum_{i=1}^{S} w_i \cdot x_i + w_0.$$

The real valued coefficients $w_i$ are commonly referred to as the *weights* of the sigmoidal function. The case that is of most interest to us is when the inputs are binary, i.e., $X \in \{1, -1\}^n$. We shall refer to this model as *sigmoidal network*.

Another common feedforward multilayer model is one in which each basic processing unit computes a binary *linear threshold function* $sgn(F(X))$, where $F(X)$ is the same as above, and

$$sgn(F(X)) = \begin{cases} 1 & \text{if } F(X) \geq 0 \\ -1 & \text{if } F(X) < 0 \end{cases}$$

This model is often called the *threshold circuit* in the literature and recently has been studied intensively in the field of computer science.

The *size* of a network/circuit is the number of elements. The *depth* of a network/circuit is the longest path from any input gate to the output gates. We can arrange the gates in layers so that all gates in the same layer compute concurrently. (A single element can be considered as a one-layer network.) Each layer costs a unit delay in the computation. The depth of the network (which is the number of layers) can therefore be interpreted as the time for (parallel) computation.

It has been established that threshold circuit is a very powerful model of computation. Many functions of common interest such as multiplication, division and sorting can be computed in polynomial-size threshold circuits of small constant depth [19, 18, 21]. While many upper bound results for threshold circuits are known in the literature, lower bound results have only been established for restricted cases of threshold circuits. Most of the existing lower bound techniques [10, 17, 16] apply only to depth-2 threshold circuits. In [16], novel techniques which utilized analytical tools from the theory of rational approximation were developed to obtain lower bounds on the size of depth-2 threshold circuits that compute the parity function. In [20], we generalized the methods of rational approximation and our earlier techniques based on harmonic analysis to obtain the first known almost tight lower bounds on the size of threshold circuits with depth more than two. In this paper, the techniques are further generalized to yield almost tight lower bounds on the size of a more general class of neural networks in which each element computes a continuous function.

The presentation of this paper will be divided into two parts. In the first part, we shall focus on results concerning threshold circuits. In the second part, the lower bound results presented in the first part are generalized and shown to be valid even when the elements of the networks can assume continuous output values. The class of networks for which such techniques can be applied include networks of sigmoidal elements and radial basis elements. Due to space limitations, we shall only state some of the important results; further results and detailed proofs will appear in an extended paper.

Before we present our main results, we shall give formal definitions of the neural network models and introduce some of the Boolean functions, which will be used to explore the computational power of the various networks. To present our results in a coherent fashion, we define throughout this paper a *Boolean function* as $f : \{1, -1\}^n \to \{1, -1\}$, instead of using the usual $\{0, 1\}$ notation.

**Definition 1**     A *threshold circuit* is a Boolean circuit in which every gate computes a linear threshold function with an additional property: *the weights are integers all bounded by a polynomial in n.*     □

**Remark 1**     The assumption that the weights in the threshold circuits are integers bounded by a polynomial is common in the literature. In fact, the best known lower bound result on depth-2 threshold circuit [10] does not apply to the case where exponentially large weights are allowed. On the other hand, such assumption does not pose any restriction as far as constant-depth and polynomial-size is concerned. In other words, the class of constant-depth polynomial-size threshold circuits ($TC^o$) remains the same when the weights are allowed to be arbitrary. This result was implicit in [4] and was improved in [18] by showing that any depth-$d$ threshold circuit

with arbitrary weights can be simulated by a depth-$(2d + 1)$ threshold circuit of polynomially bounded weights at the expense of a polynomial increase in size. More recently, it has been shown that any polynomial-size depth-$d$ threshold circuit with arbitrary weights can be simulated by a polynomial-size depth-$(2d + 1)$ threshold circuit. $\qquad\square$

In addition to Boolean circuits, we shall also be interested in the computation of Boolean functions by networks of continuous-valued elements. To formalize this notion, we adopt the following definitions [12]:

**Definition 2**    Let $\gamma : \mathbf{R} \to \mathbf{R}$. A $\gamma$ element with weights $w_1, ..., w_m \in \mathbf{R}$ and threshold $t$ is defined to be an element that computes the function $\gamma(\sum_{i=1}^{m} w_i x_i - t)$ where $(x_1, ..., x_m)$ is the input. A $\gamma$-network is a feedforward network of $\gamma$ elements with an additional property: *the weights $w_i$ are all bounded by a polynomial in $n$.*
$\square$

For example, when $\gamma$ is the sigmoidal function $\sigma(x)$, then we have a sigmoidal network, a common model of neural network. In fact, a threshold circuit can also be viewed as a special case of $\gamma$ network where $\gamma$ is the *sgn* function.

**Definition 3**    A $\gamma$-network $C$ is said to compute a Boolean function $f :$ $\{1, -1\}^n \to \{1, -1\}$ with separation $\epsilon > 0$ if there is some $t_C \in \mathbf{R}$ such that for any input $X = (x_1, ..., x_m)$ to the network $C$, the output element of $C$ outputs a value $C(X)$ with the following property: If $f(X) = 1$, then $C(X) \geq t_C + \epsilon$. If $f(X) = -1$, then $C(X) \leq t_C - \epsilon$. $\qquad\square$

**Remark 2**    As pointed out in [12], computing with $\gamma$ networks without separation at the output element is less interesting because an infinitesimal change in the output of any $\gamma$ element may change the output bit. In this paper, we shall be mainly interested in computations on $\gamma$ networks $C_n$ with separation at least $\Omega(n^{-k})$ for some fixed $k > 0$. This together with the assumption of polynomially bounded weights makes the complexity class of constant-depth polynomial-size $\gamma$ networks quite *robust* and more interesting to study from a theoretical point of view (see [12]). $\qquad\square$

**Definition 4**    The PARITY function of $X = (x_1, x_2, \ldots, x_n) \in \{1, -1\}^n$ is defined to be $-1$ if the number of $-1$ in the variables $x_1, ..., x_n$ is odd and $+1$ otherwise. Note that this function can be represented as the product $\prod_{i=1}^{n} x_i$. $\qquad\square$

**Definition 5**    The Complete Quadratic (CQ) function [3] is defined to be the following:
$$CQ(X) = (x_1 \wedge x_2) \oplus (x_1 \wedge x_3) \oplus \ldots \oplus (x_{n-1} \wedge x_n)$$
i.e. $CQ(X)$ is the sum modulo 2 of all AND's between the $\binom{n}{2}$ pairs of distinct variables. Note that it is also a symmetric function. $\qquad\square$

## 2    Results for Threshold Circuits

For the lower bound results on threshold circuits, a central idea of our proof is the use of a result from the theory of rational approximation which states the following

[9]: *the function sgn(x) can be approximated with an error of $O(e^{-ck/\log(1/\epsilon)})$ by a rational function of degree k for $0 < \epsilon < |x| < 1$.* (In [16], they apply an equivalent result [15] that gives an approximation to the function $|x|$ instead of $sgn(x)$.) This result allows us to approximate several layers of threshold gates by a rational function of low (i.e. logarithmic) degree when the size of the circuit is small. Then by upper bounding the degree of the rational function that approximates the PARITY function, we give a lower bound on the size of the circuit. We also give similar lower bound on the Complete Quadratic (CQ) function using the same degree argument. By generalizing the 'telescoping' techniques in [14], we show an almost matching upper bound on the size of the circuits computing the PARITY and the CQ functions. We also examine circuits in which additional gates other than the threshold gates are allowed and generalize the lower bound results in this model. For this purpose, we introduce tools from harmonic analysis of Boolean functions [11, 3, 18, 17]. We define the class of functions called $\widetilde{SP}$ such that every function in $\widetilde{SP}$ can be closely approximated by a *sparse polynomial* for all inputs. For example, it can be shown that [18] the class $\widetilde{SP}$ contains functions AND, OR, COMPARISON and ADDITION, and more generally, functions that have polynomially bounded spectral norms.

The main results on threshold circuits can be summarized by the following theorems. First we present an explicit construction for implementing PARITY. This construction applies to any 'periodic' symmetric function, such as the CQ function.

**Theorem 1**    For every $d < \log n$, there exists a depth-$(d+1)$ threshold circuit with $O(dn^{1/d})$ gates that computes the PARITY function.    □

We next show that any depth-$(d+1)$ threshold circuit computing the PARITY function or the CQ function must have size $\Omega(dn^{1/d-\epsilon})$ for any fixed $\epsilon > 0$. This result also holds for any function that has strong degree $\Omega(n)$.

**Theorem 2**    Any depth-$(d+1)$ threshold circuit computing the PARITY (CQ) function must have size $\Omega(dn^{1/d}/\log^2 n)$.    □

We also consider threshold circuits that *approximate* the PARITY and the CQ functions when we have *random* inputs which are uniformly distributed. We derive *almost tight* upper and lower bounds on the size of the approximating threshold circuits.

We next consider threshold circuits with additional gates and prove the following result.

**Theorem 3**    Suppose in addition to threshold gates, we have polynomially many gates $\in \widetilde{SP}$ in the first layer of a depth-2 threshold circuit that computes the CQ function. Then the number of threshold gates required in the circuit is $\Omega(n/\log^2 n)$. □

This result can be extended to higher depth circuits when additional gates that have *low degree polynomial approximations* are allowed.

**Remark 3**    Recently Beigel [2], using techniques similar to ours and the fact

that the PARITY function cannot be computed in polynomial-size constant-depth circuits of AND, OR gates [7], has shown that any constant-depth threshold circuit with $(2^{n^{o(1)}})$ AND, OR gates but only $o(\log n)$ threshold gates cannot compute the *PARITY* function of $n$ variables.                                                              □

## 3   Results for $\gamma$-Networks

In the second part of the paper, we consider the computational power of networks of continuous-output elements. A celebrated result in this area was obtained by Cybenko [5]. It was shown in [5] that any continuous function over a compact domain can be closely approximated by sigmoidal networks with two layers. More recently, Barron [1] has significantly strengthened this result by showing that a wide class of functions can be approximated with mean squared error of $O(n^{-1})$ by two-layer sigmoidal networks of only $n$ elements. Here we are interested in networks of continuous-output elements computing Boolean functions instead of continuous functions. See Section 1.1 for a precise definition of computation of Boolean functions by a $\gamma$-network.

While quite a few techniques have been developed for deriving lower bound results on the complexity of threshold circuits, an understanding of the power and the limitation of networks of continuous elements such as sigmoidal networks, especially as compared to threshold circuits, have not been explored. For example, we would like to answer questions such as: how much added computational power does one gain by using sigmoidal elements or other continuous elements to compute *Boolean* functions? Can the size of the network be reduced by using sigmoidal elements instead of threshold elements?

It was shown in [12] when the depth of the network is restricted to be two, then there is a Boolean function of $n$ variables that can be computed in a depth-2 sigmoidal network with a fixed number of elements, but requires a depth-2 threshold circuit with size that increases at least logarithmic in $n$. In other words, in the restricted case of depth-2 network, one can reduce the size of the network *at least* a logarithmic factor by using continuous elements such as the sigmoidal elements instead of threshold elements with binary output values. This result has been recently improved in [6], where it is shown that there exists an explicit function that can be computed using only a constant number of sigmoidal gates, and that *any* threshold circuit (irrespective of the depth) computing it must have size $\Omega(\log n)$.

These results motivate the following question: Can we characterize a class of functions for which the threshold circuits computing the functions have sizes *at most* a logarithmic factor larger than the sizes of the sigmoidal networks computing them? Because of the monotonicity of the sigmoidal functions, we do not expect that there is substantial gain in the computational power over the threshold elements for computing the class of highly oscillating functions.

It is natural to extend our techniques to sigmoidal networks by approximating sigmoidal functions with rational functions. We derive a key lemma that yields a single low degree rational approximation to any function that can be *piecewise approximated* by low degree rational functions.

**Lemma 1**    Let $f$ be a continuous function over $\Delta = [a,b]$. Let $\Delta_1 = [a,c]$ and $\Delta_2 = [c,b]$, $a < c < b$. Denote $\| g \|_{\Delta_i} = \sup_{x \in \Delta_i} |g(x)|$. Suppose there are rational functions $r_1$ and $r_2$ such that

$$\| f - r_i \|_{\Delta_i} \leq \epsilon$$

where $\epsilon > 0$. Then for each $\tilde{\epsilon} > 0$ and $\delta > 0$, there is a rational function $r$ such that

$$\| f - r \|_{\Delta} \leq \epsilon + \tilde{\epsilon} + \omega(f;\delta)_{\Delta}$$

$$\deg r \leq 2 \deg r_1 + 2 \deg r_2 + C_1 \log(e + \frac{b-a}{\delta}) \log(e + \frac{\| f \|_{\Delta}}{\tilde{\epsilon}}) \qquad (1)$$

where $\omega(f;\delta)_{\Delta}$ is the modulus of continuity of $f$ over $\Delta$, $C_1$ is a constant.    □

The above lemma is applied to show that both sigmoidal functions and radial basis functions can be closely approximated by low degree rational functions. In fact the above lemma can be generalized to show that if a continuous function can be piecewise approximated by low degree rational functions over $k = \log^{O(1)} n$ consecutive intervals, then it can be approximated by a *single* low degree rational function over the union of these intervals.

These generalized approximation results enable us to show that many of our lower bound results on threshold circuits can be carried over to sigmoidal networks. Prior to our work, there was no nontrivial lower bound on the size of sigmoidal networks with depth more than two. In fact, we can generalize our results to neural networks whose elements can be *piecewise approximated* by low degree rational functions. We show in this paper that for symmetric Boolean functions of large strong degree (e.g. the parity function), any depth-$d$ network whose elements can be piecewise approximated by low degree rational functions requires almost the same size as a depth-$d$ threshold circuit computing the function.

In particular, if $\tilde{R}$ is the class of polynomially bounded functions that are piecewise continuous and can be piecewise approximated with low degree rational functions, then we prove the following theorem.

**Theorem 4**    Let $W$ be any depth-$(d+1)$ neural network in which each element $v_j$ computes a function $f^j(\sum_i w_i x_i)$ where $f^j \in \tilde{R}$ and $\sum_i |w_i| \leq n^{O(1)}$ for each element. If the network $W$ computes the PARITY function of $n$ variables with separation $\delta$, where $0 < \delta = \Omega(n^{-k})$ for some $k > 0$, then for any fixed $\epsilon > 0$, $W$ must have size $\Omega(dn^{1/d-\epsilon})$.    □

# References

[1] A. Barron. Universal Approximation Bounds for Superpositions of a Sigmoidal Function . *IEEE Transactions on Information Theory*, to appear.

[2] R. Beigel. Polylog($n$) Majority or $O(\log \log n)$ Symmetric Gates are Equivalent to One. ACM Symposium on Theory of Computing (STOC), 1992.

[3] J. Bruck. Harmonic Analysis of Polynomial Threshold Functions . *SIAM Journal on Discrete Mathematics*, pages 168–177, May 1990.

[4] A. K. Chandra, L. Stockmeyer, and U. Vishkin. Constant depth reducibility. *Siam J. Comput.*, 13:423–439, 1984.

[5] G. Cybenko. Approximations by superpositions of a sigmoidal function. *Math. Control, Signals, Systems*, vol. 2, pages 303–314, 1989.

[6] B. Dasgupta and G. Schnitger. Efficient Approximation with Neural Networks: A Comparison of Gate Functions. In 5th Annual Conference on Neural Information Processing Systems - Natural and Synthetic (NIPS'92), 1992.

[7] M. Furst, J. B. Saxe, and M. Sipser. Parity, Circuits and the Polynomial-Time Hierarchy. *IEEE Symp. Found. Comp. Sci.*, 22:260–270, 1981.

[8] M. Goldmann, J. Håstad, and A. Razborov. Majority Gates vs. General Weighted Threshold Gates. *Seventh Annual Conference on Structure in Complexity Theory, 1992.*

[9] A. A. Gončar. On the rapidity of rational approximation of continuous functions with characteristic singularities. *Mat. Sbornik*, 2(4):561–568, 1967.

[10] A. Hajnal, W. Maass, P. Pudlak, M. Szegedy, and G. Turan. Threshold circuits of bounded depth. *IEEE Symp. Found. Comp. Sci.*, 28:99–110, 1987.

[11] R. J. Lechner. *Harmonic analysis of switching functions.* In A. Mukhopadhyay, editor, Recent Development in Switching Theory. Academic Press, 1971.

[12] W. Maass, G. Schnitger, and E. Sontag. On the computational power of sigmoid versus boolean threshold circuits. IEEE Symp. Found. Comp. Sci., October 1991.

[13] J. L. McClelland D. E. Rumelhart and the PDP Research Group. *Parallel Distributed Processing: Explorations in the Microstructure of Cognition, vol. 1.* MIT Press, 1986.

[14] R. Minnick. Linear-Input Logic. *IEEE Trans. on Electronic Computers*, EC 10, 1961.

[15] D. J. Newman. Rational Approximation to $|x|$. *Michigan Math. Journal*, 11:11–14, 1964.

[16] R. Paturi and M. Saks. On Threshold Circuits for Parity. IEEE Symp. Found. Comp. Sci., October 1990.

[17] V. P. Roychowdhury, K. Y. Siu, A. Orlitsky, and T. Kailath. A Geometric Approach to Threshold Circuit Complexity . Workshop on Computational Learning Theory (Colt'91), pp. 97–111, 1991.

[18] K. Y. Siu and J. Bruck. On the Power of Threshold Circuits with Small Weights . SIAM J. Discrete Math, pp. 423-435, August 1991.

[19] K. Y. Siu and J. Bruck. Neural Computation of Arithmetic Functions. Proceedings of the IEEE, Special Issue on Neural Networks, pp. 1669–1675, October 1990.

[20] K. Y. Siu, V. P. Roychowdhury, and T. Kailath. Computing with Almost Optimal Size Threshold Circuits . *IEEE International Symposium on Information Theory, Budapest, Hungary*, June 1991.

[21] K.-Y. Siu, J. Bruck, T. Kailath, and T. Hofmeister. Depth-Efficient Neural Networks for Division and Related Problems . to appear in *IEEE Trans. Information Theory*, 1993.
